# Robust Novelty Detection with Single-Class MPM

**Gert R.G. Lanckriet**
EECS, U.C. Berkeley
*gert@eecs.berkeley.edu*

**Laurent El Ghaoui**
EECS, U.C. Berkeley
*elghaoui@eecs.berkeley.edu*

**Michael I. Jordan**
Computer Science and
Statistics, U.C. Berkeley
*jordan@cs.berkeley.edu*

## Abstract

In this paper we consider the problem of novelty detection, presenting an algorithm that aims to find a minimal region in input space containing a fraction $\alpha$ of the probability mass underlying a data set. This algorithm—the "single-class minimax probability machine (MPM)"—is built on a distribution-free methodology that minimizes the worst-case probability of a data point falling outside of a convex set, given only the mean and covariance matrix of the distribution and making no further distributional assumptions. We present a robust approach to estimating the mean and covariance matrix within the general two-class MPM setting, and show how this approach specializes to the single-class problem. We provide empirical results comparing the single-class MPM to the single-class SVM and a two-class SVM method.

## 1 Introduction

*Novelty detection* is an important unsupervised learning problem in which test data are to be judged as having been generated from the same or a different process as that which generated the training data. In essence, we wish to estimate a *quantile* of the distribution underlying the training data: for a fixed constant $\alpha \in (0, 1]$, we attempt to find a (small) set $\mathcal{Q}$ such that $\mathbf{Pr}\{\mathbf{y} \in \mathcal{Q}\} = \alpha$, where, for novelty detection, $\alpha$ is typically chosen near one (Schölkopf and Smola, 2001, Ben-David and Lindenbaum, 1997). This formulation of novelty detection in terms of quantile estimation is to be compared to the (costly) approach of estimating a density based on the training data and thresholding the estimated density.

Although of reduced complexity when compared to density estimation, multivariate quantile estimation is still a challenging problem, necessitating computationally efficient methods for representing and manipulating sets in high dimensions. A significant step forward in this regard was provided by Schölkopf and Smola (2001), who treated novelty detection as a "single-class" classification problem in which data are separated from the origin in feature space. This allowed them to invoke the computationally-efficient technology of support vector machines.

In the current paper we adopt the "single-class" perspective of Schölkopf and Smola (2001), but make use of a different kernel-based technique for finding discriminant

boundaries—the minimax probability machine (MPM) of Lanckriet et al. (2002). To see why the MPM should be particularly appropriate for quantile estimation, consider the following theorem, which lies at the core of the MPM. Given a random vector $\mathbf{y}$ with mean $\bar{\mathbf{y}}$ and covariance matrix $\Sigma_y$, and given arbitrary constants $\mathbf{a} \neq 0, b$ such that $\mathbf{a}^T \bar{\mathbf{y}} \leq b$, we have (for a proof, see Lanckriet et al., 2002):

$$\inf_{\mathbf{y} \sim (\bar{\mathbf{y}}, \Sigma_{\mathbf{y}})} \mathbf{Pr}\{\mathbf{a}^T \mathbf{y} \leq b\} \geq \alpha \quad \Leftrightarrow \quad b - \mathbf{a}^T \bar{\mathbf{y}} \geq \kappa(\alpha) \sqrt{\mathbf{a}^T \Sigma_{\mathbf{y}} \mathbf{a}}, \tag{1}$$

where $\kappa(\alpha) = \sqrt{\alpha/1 - \alpha}$, and $\alpha \in [0, 1)$. Note that this is a "distribution-free" result—the infimum is taken over all distributions for $\mathbf{y}$ having mean $\bar{\mathbf{y}}$ and covariance matrix $\Sigma_{\mathbf{y}}$ (assumed to be positive definite for simplicity). While Lanckriet et al. (2002) were able to exploit this theorem to design a binary classification algorithm, it is clear that the theorem provides even more direct leverage on the "single-class" problem—it directly bounds the probability of an observation falling outside of a given set.

There is one important aspect of the MPM formulation that needs further consideration, however, if we wish to apply the approach to the novelty detection problem. In particular, $\bar{\mathbf{y}}$ and $\Sigma_y$ are usually unknown in practice and must be estimated from data. In the classification setting, Lanckriet et al. (2002) successfully made use of plug-in estimates of these quantities—in some sense the bias incurred by the use of plug-in estimates in the two classes appears to "cancel" and have diminished overall impact on the discriminant boundary. In the one-class setting, however, the uncertainty due to estimation of $\bar{\mathbf{y}}$ and $\Sigma_y$ translates directly into movement of the discriminant boundary and cannot be neglected.

We begin in Section 2 by revisiting the MPM and showing how to account for uncertainty in the means and covariance matrices within the framework of robust estimation. Section 3 then applies this robust estimation approach to the single-class MPM problem. We present empirical results in Section 4 and present our conclusions in Section 5.

## 2 Robust Minimax Probability Machine (R-MPM)

Let $\mathbf{x}, \mathbf{y} \in \mathbb{R}^n$ denote random vectors in a binary classification problem, modelling data from each of two classes, with means and covariance matrices given by $\bar{\mathbf{x}}, \bar{\mathbf{y}} \in \mathbb{R}^n$, and $\Sigma_{\mathbf{x}}, \Sigma_{\mathbf{y}} \in \mathbb{R}^{n \times n}$ (both symmetric and positive semidefinite), respectively. We wish to determine a hyperplane $\mathcal{H}(\mathbf{a}, b) = \{\mathbf{z} \mid \mathbf{a}^T \mathbf{z} = b\}$, where $\mathbf{a} \in \mathbb{R}^n \backslash \{0\}$ and $b \in \mathbb{R}$, that maximizes the worst-case probability $\alpha$ that future data points are classified correctly with respect to all distributions having these means and covariance matrices:

$$\max_{\alpha, \mathbf{a} \neq 0, b} \alpha \quad \text{s.t.} \quad \inf_{\mathbf{x} \sim (\bar{\mathbf{x}}, \Sigma_{\mathbf{x}})} \mathbf{Pr}\{\mathbf{a}^T \mathbf{x} \geq b\} \geq \alpha \tag{2}$$

$$\inf_{\mathbf{y} \sim (\bar{\mathbf{y}}, \Sigma_{\mathbf{y}})} \mathbf{Pr}\{\mathbf{a}^T \mathbf{y} \leq b\} \geq \alpha,$$

where $\mathbf{x} \sim (\bar{\mathbf{x}}, \Sigma_{\mathbf{x}})$ refers to the class of distributions that have mean $\bar{\mathbf{x}}$ and covariance $\Sigma_{\mathbf{x}}$, but are otherwise arbitrary; likewise for $\mathbf{y}$. The worst-case probability of misclassification is explicitly obtained and given by $1 - \alpha$.

Solving this optimization problem involves converting the probabilistic constraints in Eq. (2) into deterministic constraints, a step which is achieved via the theorem referred to earlier in Eq. (1). This eventually leads to the following convex optimization problem, whose solution determines an optimal hyperplane $\mathcal{H}(\mathbf{a}, b)$ (Lanckriet

et al., 2002):

$$\kappa_*^{-1} := \min_{\mathbf{a}} \sqrt{\mathbf{a}^T \Sigma_\mathbf{x} \mathbf{a}} + \sqrt{\mathbf{a}^T \Sigma_\mathbf{y} \mathbf{a}} \quad \text{s.t.} \quad \mathbf{a}^T(\bar{\mathbf{x}} - \bar{\mathbf{y}}) = 1, \qquad (3)$$

where $b$ is set to the value $b_* = \mathbf{a}_*^T \bar{\mathbf{x}} - \kappa_* \sqrt{\mathbf{a}_*^T \Sigma_\mathbf{x} \mathbf{a}_*}$, with $\mathbf{a}_*$ an optimal solution of Eq. (3). The optimal worst-case misclassification probability is obtained via $1 - \alpha_* = 1/(1 + \kappa_*^2)$. Once an optimal hyperplane is found, classification of a new data point $\mathbf{z}_{new}$ is done by evaluating $\text{sign}(\mathbf{a}_*^T \mathbf{z}_{new} - b_*)$: if this is $+1$, $\mathbf{z}_{new}$ is classified as belonging to class $\mathbf{x}$, otherwise $\mathbf{z}_{new}$ is classified as belonging to class $\mathbf{y}$.

While in our earlier work, we simply computed sample-based estimates of means and covariance matrices and plugged them into the MPM optimization problem in Eq. (3), we now show how to treat this estimation problem within the framework of robust optimization. Assume the mean and covariance matrix of each class are unknown but lie within specified convex sets: $(\bar{\mathbf{x}}, \Sigma_\mathbf{x}) \in \mathcal{X}$, with $\mathcal{X} \subset \mathbb{R}^n \times \{M \in \mathbb{R}^{n \times n} | M = M^T, M \succeq 0\}$, and $(\bar{\mathbf{y}}, \Sigma_\mathbf{y}) \in \mathcal{Y}$, with $\mathcal{Y} \subset \mathbb{R}^n \times \{M \in \mathbb{R}^{n \times n} | M = M^T, M \succeq 0\}$. We now want the probabilistic guarantees in Eq. (2) to be robust against variations of the mean and covariance matrix within these sets:

$$\max_{\alpha, \mathbf{a} \neq 0, b} \alpha \quad \text{s.t.} \quad \inf_{\mathbf{x} \sim (\bar{\mathbf{x}}, \Sigma_\mathbf{x})} \mathbf{Pr}\{\mathbf{a}^T \mathbf{x} \geq b\} \geq \alpha \ \forall (\bar{\mathbf{x}}, \Sigma_\mathbf{x}) \in \mathcal{X}, \qquad (4)$$

$$\inf_{\mathbf{x} \sim (\bar{\mathbf{y}}, \Sigma_\mathbf{y})} \mathbf{Pr}\{\mathbf{a}^T \mathbf{y} \leq b\} \geq \alpha \ \forall (\bar{\mathbf{y}}, \Sigma_\mathbf{y}) \in \mathcal{Y}.$$

In other words, we would like to guarantee a worst-case misclassification probability for all distributions which have unknown-but-bounded mean and covariance matrix, but which are otherwise arbitrary. The complexity of this problem depends obviously on the structure of the uncertainty sets $\mathcal{X}, \mathcal{Y}$. We now consider a specific choice for $\mathcal{X}$ and $\mathcal{Y}$, motivated both statistically and numerically:

$$
\begin{aligned}
\mathcal{X} &= \left\{ (\bar{\mathbf{x}}, \Sigma_\mathbf{x}) : (\bar{\mathbf{x}} - \bar{\mathbf{x}}^0)^T \Sigma_\mathbf{x}^{-1} (\bar{\mathbf{x}} - \bar{\mathbf{x}}^0) \leq \nu^2, \ \|\Sigma_\mathbf{x} - \Sigma_\mathbf{x}^0\|_F \leq \rho \right\}, \\
\mathcal{Y} &= \left\{ (\bar{\mathbf{y}}, \Sigma_\mathbf{y}) : (\bar{\mathbf{y}} - \bar{\mathbf{y}}^0)^T \Sigma_\mathbf{y}^{-1} (\bar{\mathbf{y}} - \bar{\mathbf{y}}^0) \leq \nu^2, \ \|\Sigma_\mathbf{y} - \Sigma_\mathbf{y}^0\|_F \leq \rho \right\},
\end{aligned} \qquad (5)
$$

with $\bar{\mathbf{x}}^0, \Sigma_\mathbf{x}^0$ the "nominal" mean and covariance estimates and with $\nu, \rho \geq 0$ fixed and, for simplicity, assumed equal for $\mathcal{X}$ and $\mathcal{Y}$. Section 4 discusses how their values can be determined. The matrix norm is the Frobenius norm: $\|A\|_F^2 = \mathbf{Tr}(A^T A)$.

Our model for the uncertainty in the mean assumes the mean of class $\mathbf{y}$ belongs to an ellipsoid — a convex set — centered around $\bar{\mathbf{y}}^0$, with shape determined by the (unknown) $\Sigma_\mathbf{y}$. This is motivated by the standard statistical approach to estimating a region of confidence based on Laplace approximations to a likelihood function. The covariance matrix belongs to a matrix norm ball — a convex set — centered around $\Sigma_\mathbf{y}^0$. This uncertainty model is perhaps less classical from a statistical viewpoint, but it will lead to a regularization term of a classical form.

In order to solve Eq. (4), we apply Eq. (1) and notice that

$$b - \mathbf{a}^T \bar{\mathbf{y}} \geq \kappa(\alpha)\sqrt{\mathbf{a}^T \Sigma_\mathbf{y} \mathbf{a}}, \forall (\bar{\mathbf{y}}, \Sigma_\mathbf{y}) \in \mathcal{Y} \Leftrightarrow b - \max_{(\bar{\mathbf{y}}, \Sigma_\mathbf{y}) \in \mathcal{Y}} \mathbf{a}^T \bar{\mathbf{y}} \geq \kappa(\alpha)\sqrt{\max_{(\bar{\mathbf{y}}, \Sigma_\mathbf{y}) \in \mathcal{Y}} \mathbf{a}^T \Sigma_\mathbf{y} \mathbf{a}},$$

where the right-hand side guarantees the constraint for the worst-case estimate of the mean and covariance matrix within the bounded set $\mathcal{Y}$. For given $\mathbf{a}$ and $\bar{\mathbf{y}}^0$:

$$\min_{\bar{\mathbf{y}} \, : \, (\bar{\mathbf{y}} - \bar{\mathbf{y}}^0)^T \Sigma_\mathbf{y}^{-1}(\bar{\mathbf{y}} - \bar{\mathbf{y}}^0) \leq \nu^2} -\mathbf{a}^T \bar{\mathbf{y}} = -\mathbf{a}^T \bar{\mathbf{y}}^0 - \nu\sqrt{\mathbf{a}^T \Sigma_\mathbf{y} \mathbf{a}}. \qquad (6)$$

Indeed, the Lagrangian is $\mathcal{L}(\bar{\mathbf{y}}, \lambda) = -\mathbf{a}^T \bar{\mathbf{y}} + \lambda((\bar{\mathbf{y}} - \bar{\mathbf{y}}^0)^T \Sigma_\mathbf{y}^{-1}(\bar{\mathbf{y}} - \bar{\mathbf{y}}^0) - \nu^2)$ and is to be maximized with respect to $\lambda \geq 0$ and minimized with respect to $\bar{\mathbf{y}}$. At the

optimum, we have $\frac{\partial}{\partial \bar{\mathbf{y}}} \mathcal{L}(\bar{\mathbf{y}}, \lambda) = 0$ and $\frac{\partial}{\partial \lambda} \mathcal{L}(\bar{\mathbf{y}}, \lambda) = 0$, leading to $\bar{\mathbf{y}} = \bar{\mathbf{y}}^0 + \frac{1}{2\lambda} \boldsymbol{\Sigma}_{\mathbf{y}} \mathbf{a}$ and $\lambda = \sqrt{\mathbf{a}^T \boldsymbol{\Sigma}_{\mathbf{y}} \mathbf{a} / 4\nu}$ which eventually leads to Eq. (6). For given $\mathbf{a}$ and $\boldsymbol{\Sigma}_{\mathbf{y}}^0$:

$$\max_{\boldsymbol{\Sigma}_{\mathbf{y}} \,:\, \|\boldsymbol{\Sigma}_{\mathbf{y}} - \boldsymbol{\Sigma}_{\mathbf{y}}^0\|_F \leq \rho} \mathbf{a}^T \boldsymbol{\Sigma}_{\mathbf{y}} \mathbf{a} = \mathbf{a}^T \left( \boldsymbol{\Sigma}_{\mathbf{y}}^0 + \rho I_n \right) \mathbf{a}, \tag{7}$$

where $I_n$ is the $n \times n$ identity matrix. Indeed, without loss of generality, we can let $\boldsymbol{\Sigma}$ be of the form $\boldsymbol{\Sigma} = \boldsymbol{\Sigma}^0 + \rho \Delta \boldsymbol{\Sigma}$. We then obtain

$$\max_{\boldsymbol{\Sigma}_{\mathbf{y}} \,:\, \|\boldsymbol{\Sigma}_{\mathbf{y}} - \boldsymbol{\Sigma}_{\mathbf{y}}^0\|_F \leq \rho} \mathbf{a}^T \boldsymbol{\Sigma}_{\mathbf{y}} \mathbf{a} = \mathbf{a}^T \boldsymbol{\Sigma}_{\mathbf{y}}^0 \mathbf{a} + \rho \max_{\Delta \boldsymbol{\Sigma}_{\mathbf{y}} \,:\, \|\Delta \boldsymbol{\Sigma}_{\mathbf{y}}\|_F \leq 1} \mathbf{a}^T \Delta \boldsymbol{\Sigma}_{\mathbf{y}} \mathbf{a} = \mathbf{a}^T \boldsymbol{\Sigma}_{\mathbf{y}}^0 \mathbf{a} + \rho \mathbf{a}^T \mathbf{a},$$
$$\tag{8}$$

using the Cauchy-Schwarz inequality and compatibility of the Frobenius matrix norm and the Euclidean vector norm:

$$\mathbf{a}^T \Delta \boldsymbol{\Sigma} \mathbf{a} \leq \|\mathbf{a}\|_2 \|\Delta \boldsymbol{\Sigma} \mathbf{a}\|_2 \leq \|\mathbf{a}\|_2 \|\Delta \boldsymbol{\Sigma}\|_F \|\mathbf{a}\|_2 \leq \|\mathbf{a}\|_2^2,$$

because $\|\Delta \boldsymbol{\Sigma}\|_F \leq 1$. For $\Delta \boldsymbol{\Sigma} = I_n$, this upper bound is attained and we get Eq. (7). Combining this with Eq. (6) leads to the robust version of Eq. (1):

$$\inf_{\mathbf{y} \sim (\bar{\mathbf{y}}, \boldsymbol{\Sigma}_{\mathbf{y}})} \mathbf{Pr}\{\mathbf{a}^T \mathbf{y} \leq b\} \geq \alpha, \ \forall (\bar{\mathbf{y}}, \boldsymbol{\Sigma}_{\mathbf{y}}) \in \mathcal{Y} \Leftrightarrow b - \mathbf{a}^T \bar{\mathbf{y}}^0 \geq (\kappa(\alpha) + \nu) \sqrt{\mathbf{a}^T (\boldsymbol{\Sigma}_{\mathbf{y}}^0 + \rho I_n) \mathbf{a}}.$$
$$\tag{9}$$

Applying this result to Eq. (4) thus shows that the optimal robust minimax probability classifier for $\mathcal{X}, \mathcal{Y}$ given by Eq. (5) can be obtained by solving problem Eq. (3), with $\boldsymbol{\Sigma}_{\mathbf{x}} = \boldsymbol{\Sigma}_{\mathbf{x}}^0 + \rho I_n$, $\boldsymbol{\Sigma}_{\mathbf{y}} = \boldsymbol{\Sigma}_{\mathbf{y}}^0 + \rho I_n$. If $\kappa_*^{-1}$ is the optimal value of that problem, the corresponding worst-case misclassification probability is

$$1 - \alpha_* = \frac{1}{1 + \max(0, (\kappa_* - \nu))^2}.$$

With only uncertainty in the mean ($\rho = 0$), the robust hyperplane is the *same* as the non-robust one; the only change is in the increase in the worst-case misclassification probability. Uncertainty in the covariance matrix adds a term $\rho I_n$ to the covariance matrices, which can be interpreted as regularization term. This affects the hyperplane and increases the worst-case misclassification probability as well. If there is too much uncertainty in the mean (i.e., $\kappa_* < \nu$), the robust version is not feasible: no hyperplane can be found that separates the two classes in the robust minimax probabilistic sense and the worst-case misclassification probability is $1 - \alpha_* = 1$.

This robust approach can be readily generalized to allow nonlinear decision boundaries via the use of Mercer kernels (Lanckriet et al., 2002).

## 3  Single-class MPM for robust novelty detection

We now turn to the quantile estimation problem. Recall that for $\alpha \in (0, 1]$, we wish to find a small region $\mathcal{Q}$ such that $\mathbf{Pr}\{\mathbf{x} \in \mathcal{Q}\} = \alpha$. Let us consider data $\mathbf{x} \sim (\bar{\mathbf{x}}, \boldsymbol{\Sigma}_{\mathbf{x}})$ and let us focus (for now) on the linear case where $\mathcal{Q}$ is a half-space not containing the origin.

We seek a half-space $\mathcal{Q}(\mathbf{a}, b) = \{\mathbf{z} \mid \mathbf{a}^T \mathbf{z} \geq b\}$, with $\mathbf{a} \in \mathbb{R}^n \backslash \{0\}$ and $b \in \mathbb{R}$, and not containing $\mathbf{0}$, such that with probability at least $\alpha$, the data lies in $\mathcal{Q}$, for every distribution having mean $\bar{\mathbf{x}}$ and covariance matrix $\boldsymbol{\Sigma}_{\mathbf{x}}$. We assume again that the real $\bar{\mathbf{x}}, \boldsymbol{\Sigma}_{\mathbf{x}}$ are unknown but bounded in a set $\mathcal{X}$ as specified in Eq. (5):

$$\inf_{\mathbf{x} \sim (\bar{\mathbf{x}}, \boldsymbol{\Sigma}_{\mathbf{x}})} \mathbf{Pr}\{\mathbf{a}^T \mathbf{x} \geq b\} \geq \alpha \quad \forall (\bar{\mathbf{x}}, \boldsymbol{\Sigma}_{\mathbf{x}}) \in \mathcal{X}.$$

We want the region $\mathcal{Q}$ to be tight, so we maximize its Mahalanobis distance (with respect to $\mathbf{\Sigma_x}$) to the origin in a robust way, i.e., for the worst-case estimate of $\mathbf{\Sigma_x}$—the matrix that gives us the smallest Mahalanobis distance:

$$\max_{\mathbf{a}\neq 0, b} \min_{(\bar{\mathbf{x}}, \mathbf{\Sigma_x}) \in \mathcal{X}} \frac{b}{\sqrt{\mathbf{a}^T \mathbf{\Sigma_x} \mathbf{a}}} \quad \text{s.t.} \quad \inf_{\mathbf{x} \sim (\bar{\mathbf{x}}, \mathbf{\Sigma_x})} \mathbf{Pr}\{\mathbf{a}^T \mathbf{x} \geq b\} \geq \alpha \quad \forall (\bar{\mathbf{x}}, \mathbf{\Sigma_x}) \in \mathcal{X}. \quad (10)$$

Note that $\mathcal{Q}(\mathbf{a}, b)$ does not contain $\mathbf{0}$ if and only if $b > 0$. Also, the optimization problem in Eq. (10) is positively homogeneous in $(\mathbf{a}, b)$. Thus, without loss of generality, we can set $b = 1$ in problem Eq. (10). Furthermore, we can use Eq. (7) and Eq. (9) and get (where superscript 0 for the estimates has been omitted):

$$\min_{\mathbf{a}} \sqrt{\mathbf{a}^T (\mathbf{\Sigma_x} + \rho I_n)\mathbf{a}} \quad \text{s.t.} \quad \mathbf{a}^T \bar{\mathbf{x}} - 1 \geq (\kappa(\alpha) + \nu)\sqrt{\mathbf{a}^T (\mathbf{\Sigma_x} + \rho I_n)\mathbf{a}}, \quad (11)$$

where $\mathbf{a} \neq 0$ can be omitted since the constraint never holds in this case. Again, we obtain a (convex) second order cone programming problem. The worst-case probability of occurrence outside region $\mathcal{Q}$ is given by $1 - \alpha$. Notice that the particular choice of $\alpha \in (0, 1]$ must be feasible, i.e.,

$$\exists \mathbf{a} \; : \; \mathbf{a}^T \bar{\mathbf{x}} - 1 \geq (\kappa(\alpha) + \nu)\sqrt{\mathbf{a}^T (\mathbf{\Sigma_x} + \rho I_n)\mathbf{a}}.$$

For $\rho \neq 0$, $\mathbf{\Sigma_x} + \rho I_n$ is certainly positive definite and the halfspace is unique. Furthermore, it can be determined explicitly. To see this, we write Eq. (11) as:

$$\min_{\mathbf{a}} \; \|(\mathbf{\Sigma_x} + \rho I_n)^{1/2}\mathbf{a}\|_2 \quad \text{s.t.} \quad \mathbf{a}^T \bar{\mathbf{x}} \geq 1 + (\kappa(\alpha) + \nu)\|(\mathbf{\Sigma_x} + \rho I_n)^{1/2}\mathbf{a}\|_2 \quad (12)$$

Decomposing $\mathbf{a}$ as $\lambda(\mathbf{\Sigma_x} + \rho I_n)^{-1}\bar{\mathbf{x}} + \mathbf{z}$, where the variable $\mathbf{z}$ satisfies $\mathbf{z}^T \bar{\mathbf{x}} = 0$, we easily obtain that at the optimum, $\mathbf{z} = 0$. In other words, the optimal $\mathbf{a}$ is parallel to $\bar{\mathbf{x}}$, in the form $\mathbf{a} = \lambda(\mathbf{\Sigma_x} + \rho I_n)^{-1}\bar{\mathbf{x}}$, and the problem reduces to the one-dimensional problem:

$$\min_{\lambda} |\lambda| \; \|(\mathbf{\Sigma_x} + \rho I_n)^{-1/2}\bar{\mathbf{x}}\|_2 \; : \; \lambda \bar{\mathbf{x}}^T(\mathbf{\Sigma_x} + \rho I_n)^{-1}\bar{\mathbf{x}} \geq 1 + (\kappa(\alpha) + \nu)\|(\mathbf{\Sigma_x} + \rho I_n)^{-1/2}\bar{\mathbf{x}}\|_2 |\lambda|.$$

The constraint implies that $\lambda \geq 0$, hence the problem reduces to

$$\min_{\lambda \geq 0} \lambda \; : \; \lambda \left(\zeta^2 - (\kappa(\alpha) + \nu)\zeta\right) \geq 1. \quad (13)$$

with $\zeta^2 = \bar{\mathbf{x}}^T(\mathbf{\Sigma_x} + \rho I_n)^{-1}\bar{\mathbf{x}} > 0$ (because Eq. (12) implies $\bar{\mathbf{x}} \neq \mathbf{0}$). Because $\lambda \geq 0$, this can only be satisfied if $\zeta^2 - (\kappa(\alpha) + \nu)\zeta \geq 0$, which is nothing other than the feasibility condition for $\alpha$:

$$\zeta^2 - (\kappa(\alpha) + \nu)\zeta \geq 0 \quad \Leftrightarrow \quad \kappa(\alpha) \leq \zeta - \nu \quad \Leftrightarrow \quad \alpha \leq \frac{(\zeta - \nu)^2}{1 + (\zeta - \nu)^2}.$$

If this is fulfilled, the optimization in Eq. (13) is feasible and boils down to:

$$\min_{\lambda \geq 0} \lambda \quad \text{s.t.} \quad \lambda \geq \frac{1}{\zeta^2 - (\kappa(\alpha) + \nu)\zeta}.$$

It's easy to see that the optimal $\lambda$ is given by $\lambda_* = 1/(\zeta^2 - (\kappa(\alpha) + \nu)\zeta)$, yielding:

$$\mathbf{a}_* = \frac{(\mathbf{\Sigma_x} + \rho I_n)^{-1}\bar{\mathbf{x}}}{\zeta^2 - (\kappa(\alpha) + \nu)\zeta}, \quad b_* = 1, \quad \text{with} \quad \zeta = \sqrt{\bar{\mathbf{x}}^T(\mathbf{\Sigma_x} + \rho I_n)^{-1}\bar{\mathbf{x}}}. \quad (14)$$

Notice that the uncertainty in the covariance matrix $\mathbf{\Sigma_x}$ leads to the typical, well-known regularization for inverting this matrix. If the choice of $\alpha$ is not feasible or if $\bar{\mathbf{x}} = \mathbf{0}$ (in this case, no $\alpha \in (0, 1]$ will be feasible), Eq. (10) has no solution.

Future points $\mathbf{z}$ for which $\mathbf{a}_*^T \mathbf{z} \leq b_*$ can then be considered as outliers with respect to the region $\mathcal{Q}$, with worst-case probability of occurrence outside $\mathcal{Q}$ given by $1 - \alpha$.

One can obtain a nonlinear region $\mathcal{Q}$ in $\mathbb{R}^n$ for the single-class case, by mapping the data into a feature space $\mathbb{R}^f \colon \mathbf{x} \mapsto \varphi(\mathbf{x}) \sim (\overline{\varphi(\mathbf{x})}, \boldsymbol{\Sigma}_{\varphi(\mathbf{x})})$, and expressing and solving Eq. (10) in the feature space, using $\varphi(\mathbf{x}), \overline{\varphi(\mathbf{x})}$ and $\boldsymbol{\Sigma}_{\varphi(\mathbf{x})}$. This is achieved using a kernel function $K(\mathbf{z}_1, \mathbf{z}_2) = \varphi(\mathbf{z}_1)^T \varphi(\mathbf{z}_2)$ satisfying Mercer's condition as in the classification setting. Notice that maximizing the Mahanalobis distance of $\mathcal{Q}$ to the origin in $\mathbb{R}^f$ makes sense for novelty detection. For example, if we consider a Gaussian kernel $K(\mathbf{x}, \mathbf{y}) = e^{-\|\mathbf{x} - \mathbf{y}\|^2 / \sigma}$, all mapped data points have unit length and positive dot products, so they all lie in the same orthant, on the unit ball, and are linearly separable from the origin.

Our final result is thus the following: If the choice of $\alpha$ is feasible, i.e.,

$$\exists \gamma \ : \ \gamma^T \mathbf{k} - 1 \geq (\kappa(\alpha) + \nu)\sqrt{\gamma^T (\mathbf{L}^T \mathbf{L} + \rho \mathbf{K})\gamma},$$

then an optimal region $\mathcal{Q}(\gamma, b)$ can be determined by solving the (convex) second order cone programming problem:

$$\min_{\gamma} \ \sqrt{\gamma^T (\mathbf{L}^T \mathbf{L} + \rho \mathbf{K})\gamma} \quad \text{s.t.} \quad \gamma^T \mathbf{k} - 1 \geq (\kappa(\alpha) + \nu)\sqrt{\gamma^T (\mathbf{L}^T \mathbf{L} + \rho \mathbf{K})\gamma}, \quad (15)$$

where $\kappa(\alpha) = \sqrt{\alpha / 1 - \alpha}$ and $b = 1$, with $\gamma, \mathbf{k} \in \mathbb{R}^N$, $[\mathbf{k}]_i = \frac{1}{N}\sum_{j=1}^{N} K(\mathbf{x}_j, \mathbf{x}_i)$ and $\{\mathbf{x}_i\}_{i=1}^N$ the $N$ given data points. $\mathbf{L}$ is defined as $\mathbf{L} = (\mathbf{K} - \mathbf{1}_N \mathbf{k}^T)/\sqrt{N}$, where $\mathbf{1}_m$ is a column vector with ones of dimension $m$. $\mathbf{K}$ is the Gram matrix and defined as $\mathbf{K}_{ij} = \varphi(\mathbf{z}_i)^T \varphi(\mathbf{z}_j) = K(\mathbf{z}_i, \mathbf{z}_j)$.

The worst-case probability of a point lying outside the region $\mathcal{Q}$ is given by $1 - \alpha$. If $\mathbf{L}^T \mathbf{L} + \rho \mathbf{K}$ is positive definite, the optimal half-space is unique and determined by:

$$\gamma_* = \frac{(\mathbf{L}^T \mathbf{L} + \rho \mathbf{K})^{-1} \mathbf{k}}{\zeta^2 - (\kappa(\alpha) + \nu)\zeta} \qquad \text{with} \quad \zeta = \sqrt{\mathbf{k}^T (\mathbf{L}^T \mathbf{L} + \rho \mathbf{K})^{-1} \mathbf{k}}, \qquad (16)$$

if the choice of $\alpha$ is such that $\kappa(\alpha) \leq \zeta - \nu$ or $\alpha \leq \frac{(\zeta - \nu)^2}{1 + (\zeta - \nu)^2}$. If the choice of $\alpha$ is not feasible or if $\mathbf{k} = \mathbf{0}$ (in this case, no $\alpha \in (0, 1]$ will be feasible), the problem does not have a solution.

To solve the single-class problem, we can solve the second-order cone progam Eq. (15) or directly use result Eq. (16): when numerically regularizing $\mathbf{L}^T \mathbf{L} + \rho \mathbf{K}$ with an extra term $\epsilon I_N$, this unique solution can always be determined. Instead of explicitly inverting the matrix, we can solve a system iteratively. All of these approaches have a worst-case complexity of $O(N^3)$, comparable to the quadratic program for single-class SVM (Schölkopf and Smola, 2001).

Once an optimal decision region is found, future points $\mathbf{z}$ for which $\mathbf{a}_*^T \varphi(\mathbf{z}) = \sum_{i=1}^{N} [\gamma_*]_i K(\mathbf{x}_i, \mathbf{z}) \leq b_*$ (notice that this can be evaluated only in terms of the kernel function), can then be considered as outliers with respect to the region $\mathcal{Q}$, with the worst-case probability of occurrence outside $\mathcal{Q}$ given by $1 - \alpha$.

## 4 Experiments

In this section we report the results of experiments comparing the robust single-class MPM to the single-class SVM of Schölkopf and Smola (2001) and to a two-class SVM approach where an artificial "negative class" is obtained by generating data points uniformly in $T = \{\mathbf{z} \in \mathbb{R}^n \,|\, \min\{[\mathbf{x}_1]_i, [\mathbf{x}_2]_i, \ldots, [\mathbf{x}_N]_i\} \leq [\mathbf{z}]_i \leq \max\{[\mathbf{x}_1]_i, [\mathbf{x}_2]_i, \ldots, [\mathbf{x}_N]_i\}\}$.

For the benchmark binary classification data sets we studied, we converted the data sets into two single-class problems by treating each class in a separate experiment. We chose 80% of the data points as training and the remaining 20% of the data points as test, lumping the latter with the data points of the negative class (the class of the binary classification data, not used for training). We report false positive and false negative rates averaged over 30 random partitions in Table 1.[1]

We used a Gaussian kernel, $K(\mathbf{x}, \mathbf{y}) = e^{-\|\mathbf{x}-\mathbf{y}\|^2/\sigma}$, of width $\sigma$. The kernel parameter $\sigma$ was tuned using cross-validation over 20 random partitions, as was the hyperparameter $\rho$. For simplicity, we set the hyperparameter $\nu = 0$ for the robust single-class MPM. Note that this choice has no impact on the MPM solution; according to Eq. (16) its only effect is to alter the estimated false-negative rate.

The parameter $\alpha$ was varied throughout a range of values so as to explore the tradeoff between the false positive (FP) rate and the false negative (FN) rate. A small value $\alpha$ yields a good FP but poor FN, and large $\alpha$ yields good FN but poor FP. For the single-class SVM and the two-class SVM, we varied the analogous parameters—$\nu$ (the fraction of support vectors and outliers) and $C$ (the soft margin weight parameter)—to cover a similar range of the FP/FN tradeoff. We envision the end user deciding where he or she wishes to operate along the FP/FN tradeoff, and tuning $\alpha$, $\nu$ or $C$ accordingly. Thus we compare the different algorithms by presenting in Table 1 an overview of the full tradeoff curves (essentially the ROC curves). The specific values of $\alpha$, $\nu$ and $C$ are chosen in each row so as to roughly match corresponding points on the ROC curves. We use italic font to indicate the best performing algorithm on a given row, choosing the algorithm with the best FP rate if FN rates are similar and with the best FN rate if FP rates are similar.

The performance of the single-class MPM is clearly competitive with that of the other algorithms, providing joint FP/FN values that equal or improve upon the other algorithms in many cases, and spanning a broad range of FP/FN tradeoff. Note that the two-class SVM can perform well if low FP rate is desired and high FN rate is tolerated. However, the two-class SVM sometimes fails to provide an extensive range of FP/FN tradeoff; in particular, with the twonorm dataset, the algorithm is unable to provide solutions with small FN rate and large FP rate.

Note that the value $1-\alpha$ (the worst-case probability of false negatives for the robust single-class MPM) is indeed an upper bound for the average FN rate in all cases except for the sonar dataset. Thus the simplifying assumption $\nu = 0$ appears to be reasonable in all cases except the sonar case.

Finally, it is also worth noting that while the MPM algorithm is insensitive to the choice of $\nu$, it is sensitive to the choice of $\rho$. When we fixed $\rho = 0$ (allowing no uncertainty in the covariance estimate) we obtained poor performance, in particular obtaining a small FP rate but a very poor FN rate.

## 5   Conclusions

We have presented a new algorithm for novelty detection, an important machine learning problem with numerous real-world applications. Our "single-class MPM" joins the "single-class SVM" of Schölkopf and Smola (2001) as a computationally-efficient, kernel-based method for solving this problem and the more general quantile estimation problem. We view the single-class MPM as particularly appropriate for these problems, given its formulation directly in terms of a worst-case probability

Table 1: Performance for single-class problems; the best performance in each row is indicated in italic; FP = false positives (out-of-class data detected as in-class-data); FN = false negatives (in-class-data detected as out-of-class-data).

| Dataset | Single Class MPM | | | Single Class SVM | | | Two-Class SVM approach | | |
|---|---|---|---|---|---|---|---|---|---|
| | α | FP | FN | ν | FP | FN | C | FP | FN |
| Sonar class +1 | 0.2 | *24.7* % | *64.0* % | 0.6 | 26.9 % | 65.4 % | 0.1 | 23.8 % | 68.6 % |
| | 0.8 | *44.6* % | *39.6* % | 0.2 | 47.3 % | 42.1 % | 0.2 | 48.3 % | 42.3 % |
| | 0.95 | 69.3 % | 17.3 % | 0.0005 | 75.4 % | 16.2 % | 1 | *75.2* % | *16.0* % |
| Sonar class -1 | 0.6 | *5.4* % | 51.7 % | 0.4 | 8.5 % | 53.7 % | 0.1 | 9.7 % | 70.0 % |
| | 0.9 | *10.0* % | *37.4* % | 0.001 | 15.7 % | 41.3 % | 0.2 | 34.6 % | 40.6 % |
| | 0.95 | *19.1* % | *29.7* % | 0.0006 | 36.1 % | 28.4 % | 0.35 | 47.7 % | 26.0 % |
| | 0.99 | *56.1* % | *5.7* % | 0.0003 | 82.6 % | 6.3 % | 1 | 67.9 % | 6.1 % |
| Breast Cancer class +1 | 0.6 | *0.0* % | 8.8 % | 0.14 | 0.0 % | 14.6 % | 0.005 | 0.4 % | 8.0 % |
| | 0.8 | 1.8 % | 5.9 % | 0.001 | 2.4 % | 6.1 % | 0.1 | *0.9* % | *4.3* % |
| | 0.2 | 10.5 % | *2.7* % | 0.0003 | 11.5 % | 3.1 % | 10 | 12.3 % | 3.1 % |
| Breast Cancer class -1 | 0.01 | 2.4 % | 26.5 % | 0.4 | 2.5 % | 41.4 % | 0.8 | *0.9* % | *47.9* % |
| | 0.03 | *2.9* % | *13.5* % | 0.2 | 2.8 % | 25.0 % | 1 | 11.0 % | 45 % |
| | 0.05 | *3.0* % | *8.3* % | 0.1 | 3.1 % | 11.3 % | 2 | 89.2 % | 38.2 % |
| | 0.14 | 5.9 % | *1.9* % | 0.0005 | 9.2 % | 3.4 % | 100 | 98.0 % | 23.5 % |
| Twonorm class +1 | 0.01 | 6.3 % | 43.2 % | 0.4 | 6.2 % | 42.8 % | 0.13 | *6.8* % | *37.3* % |
| | 0.2 | 13.9 % | 22.5 % | 0.2 | *12.7* % | *22.8* % | 0.17 | 12.0 % | 24.2 % |
| | 0.4 | 22.5 % | 11.9 % | 0.0008 | 23.3 % | *9.6* % | 5 | 25.9 % | 10.5 % |
| | 0.6 | 36.9 % | 4.5 % | 0.0003 | 33.4 % | *4.5* % | | | |
| Twonorm class -1 | 0.1 | *5.6* % | *43.7* % | 0.4 | 6.0 % | 44.1 % | 0.35 | 6.1 % | 49.8 % |
| | 0.4 | *11.3* % | *23.1* % | 0.15 | 11.8 % | 24.6 % | 0.5 | 24.5 % | 23.7 % |
| | 0.6 | *16.9* % | *12.1* % | 0.0005 | 35.9 % | 12.0 % | 10 | 30.1 % | 10.0 % |
| | 0.8 | 30.1 % | *6.9* % | 0.0003 | 39.3 % | 6.9 % | | | |
| Heart class +1 | 0.46 | 13.4 % | 46.2 % | 0.4 | 13.5 % | 47.8 % | 0.05 | *11.9* % | *46.4* % |
| | 0.52 | 24.0 % | 30.9 % | 0.05 | 24.8 % | 36.7 % | 0.07 | *22.1* % | *30.3* % |
| | 0.54 | *33.5* % | *22.6* % | 0.0008 | 38.8 % | 27.0 % | 0.1 | 35.8 % | 22.9 % |
| Heart class -1 | 0.0001 | 15.9 % | 41.3 % | 0.4 | 20.8 % | 50.7 % | 0.08 | *13.9* % | *43.8* % |
| | 0.0006 | 21.2 % | 37.2 % | 0.002 | 26.3 % | 43.8 % | 0.09 | *21.0* % | *37.5* % |
| | 0.003 | *36.3* % | *27.2* % | 0.0007 | 43.7 % | 29.2 % | 0.11 | 39.2 % | 31.8 % |
| | 0.01 | *56.9* % | *15.9* % | 0.0005 | 58.4 % | 18.09 % | 0.2 | 68.6 % | 16.7 % |

of falling outside of a given convex set in feature space.

While our simulation experiments illustrate the application of generic classification techniques to the novelty detection problem—via the generation of data from an artificial "negative class" enclosing the data—we view the single-class methods as the more viable general technology. In particular, in high-dimensional problems it is difficult to specify a "negative class" in a way that yields comparable size training sets while still yielding a good characterization of a discriminant boundary.

### Acknowledgements

We acknowledge support from ONR MURI N00014-00-1-0637 and NSF grant IIS-9988642. Sincere thanks to Alex Smola for helpful conversations and suggestions.

## Footnotes

[1]The Wisconsin breast cancer dataset contained 16 missing examples which were not used. Data for the twonorm problem were generated as specified by Breiman (1997).

### References

S. Ben-David and M. Lindenbaum. Learning distributions by their density levels: A paradigm for learning without a teacher. *Journal of Computer and System Sciences*, 55: 171–182, 1997.

L. Breiman. Arcing classifiers. Technical Report Technical Report 460, Statistics Department, University of California, 1997.

G. Lanckriet, L. El Ghaoui, C. Bhattacharyya, and M. I. Jordan. A robust minimax approach to classification. *Journal of Machine Learning Research*, 3:555–582, 2002.

B. Schölkopf and A. Smola. *Learning with Kernels*. MIT Press, Cambridge, MA, 2001.
